# Speculative Monte-Carlo Tree Search

**Scott Cheng**$^{\diamond\spadesuit}$     **Mahmut Taylan Kandemir**$^{\diamond}$     **Ding-Yong Hong**$^{\spadesuit}$

$^{\diamond}$The Pennsylvania State University, USA
$^{\spadesuit}$Institute of Information Science, Academia Sinica, Taiwan

{ypc5394,mtk2}@psu.edu   dyhong@iis.sinica.edu.tw

## Abstract

Monte-Carlo tree search (MCTS) is an influential sequential decision-making algorithm notably employed in AlphaZero. Despite its success, the primary challenge in AlphaZero training lies in its prolonged time-to-solution due to the high latency imposed by the sequential MCTS process. To address this challenge, this paper proposes and evaluates an inter-decision parallelization strategy called *speculative MCTS*, a new type of parallelism in AlphaZero which implements speculative execution. This approach allows for the parallel execution of future moves *before* the current MCTS computations are completed, thus reducing the latency. Additionally, we analyze factors contributing to the overall speedup by studying the synergistic effects of speculation and neural network caching in MCTS. We also provide an analytical model that can be used to evaluate the potential of different speculation strategies before they are implemented and deployed. Our empirical findings indicate that the proposed speculative MCTS can reduce training latency by $5.81\times$ in 9x9 Go games. Moreover, our study shows that speculative execution can enhance the NN cache hit rate by 26% during midgame. Overall, our end-to-end evaluation indicates $1.91\times$ speedup in 19x19 Go training time, compared to the state-of-the-art KataGo program.

## 1  Introduction

Monte-Carlo tree search (MCTS) [1, 2] is a sequential decision-making algorithm that has achieved breakthroughs in various fields, such as chemistry [3, 4], medicine [5, 6], and many others. Most notably, AlphaZero [7] leverages MCTS to reach superhuman-level performance in "board games" such as Go, Chess, and Shogi. AlphaZero training consists of two phases: ❶ self-play and ❷ neural network (NN) training, as depicted in Figure 1. During the self-play stage, AlphaZero generates game records by playing against itself, with each move determined by MCTS simulations and NN inferences. After the self-play phase, the NN is trained on these self-play game records. By alternating between these two phases, AlphaZero continuously enhances its strategies through MCTS.

Despite AlphaZero's remarkable decision-making capabilities, the training is considerably time-consuming, and several factors contribute to long training times. First, AlphaZero training requires lots of training data per parameter, which is even higher than the well-known resource-consuming large language model (LLM) training.[1] Second, the MCTS algorithm involves *sequential* decision-making, where each game move depends on the outcome of the previous move. This sequential nature introduces significant latency. Third, the MCTS simulations evaluate the values of game positions with neural network (NN) inferences, which also contributes to the prolonged training times. In this work, we aim to *reduce* the training time by exploiting speculation-based pipeline

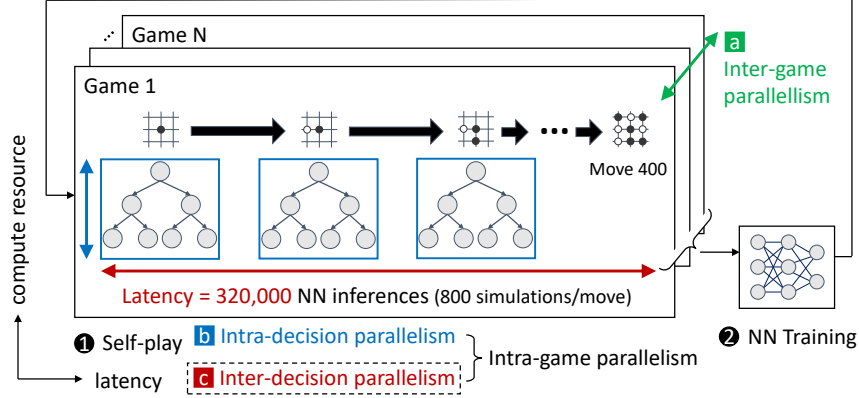

Figure 1: AlphaZero training consists of self-play and NN training, where self-play is highly resource-consuming. We categorized the intrinsic "parallelism" in the self-play phase into three categories – (a) inter-game, which is embarrassingly parallel, (b) intra-decision, such as parallel search algorithms, and (c) inter-decision, which enables parallel execution for moves within a single game. Our proposed work aims to address the problem of lack of inter-decision parallelism.

parallelism in AlphaZero training. To this end, we categorize the self-play process into three dimensions (shown in Figure 1): (a) **inter-game**, (b) **intra-decision**, and (c) **inter-decision**. Most prior works [7, 9] have focused on parallelizing "inter-game" training, as it is embarrassingly-parallel and easy to implement. Additionally, several "parallel MCTS" strategies [10, 11] have been proposed in the literature to explore and exploit intra-decision parallelism. In contrast, to the best of our knowledge, the challenge of "inter-decision" parallelism, which involves parallelizing the sequence of decisions within individual games, has *not* been addressed by prior art. As a result, even with sufficient inter-game parallelism to collect multiple game records concurrently and efficient parallel MCTS algorithms in place, the training latency remains constrained by the sequential inter-decision MCTS self-play, thus limiting the potential benefits from the increasingly powerful high-performance computing (HPC) resources and systems.

Motivated by the observations above, in this work, we propose and experimentally evaluate **speculative MCTS** to enable "inter-decision" parallelism. Since MCTS is an *Anytime Algorithm*[2] [12], it allows us to obtain "partial" simulation results (i.e., values and policies) at any point during its execution, and the simulation results keep refining as MCTS continues to execute. For example, we have observed that, more than 50% of the search results do not change after a full MCTS and only 20% of the search results would change after performing half of the MCTS simulations. Leveraging these insights, we propose to utilize the partial simulation results from the current move to *predict* the subsequent moves to play. Doing so would allow us to start executing the "future moves" before the computation of the "current move" is completed and run the MCTS computation of consecutive moves in a "pipelined fashion", thus reducing the overall training latency. Additionally, to address the NN inference latency in MCTS simulations, we integrate an "NN caching" mechanism to accelerate the retrieval of inference results. Interestingly, even for the cases where our speculation mispredicts, the NN cached results from the discarded moves can still be utilized for the current MCTS move to reduce the latency.

In summary, this paper makes the following main **contributions**:

- We design and implement a "speculative parallelization" strategy based on our insights into MCTS's anytime algorithm characteristics. We also conduct a detailed analysis of the NN cache to explore the synergy between caching and speculation. Furthermore, we conduct extensive performance evaluations by varying the number of speculative "look-ahead" steps in Go and NoGo games across different NN sizes.
- We propose an "analytical model" that allows us to estimate the performance gain and compute resource usage beforehand, which, in general, supports various designs in inter-decision parallelism. Our empirical evaluations suggest that the proposed analytical model

can accurately estimate the actual latency, with an average RMSE (root mean square error) of 22.2, across four different settings.
- Our evaluation results indicate that the proposed speculative MCTS can speed up training latency up to $5.8\times$ in 9x9 Go games. Moreover, we can achieve around 26% higher NN cache hit ratio during mid-game. Finally, our end-to-end evaluations demonstrate around $1.91\times$ speedup, compared to the state-of-the-art KataGo program.

## 2 Background

### 2.1 Monte-Carlo Tree Search

Monte Carlo Tree Search (MCTS) [1, 2] is a widely used best-first search algorithm and has demonstrated extraordinary performance when combined with neural networks in AlphaZero-like algorithms [7, 13]. Notably, KataGo [9] is the state-of-the-art, open-source computer Go program capable of defeating top-level human players. In board games, each node in the search tree represents a board position. During each simulation, MCTS selects the most promising leaf node in the search tree, and the selected node is then expanded and evaluated. During the evaluation, the NN takes the input features extracted from the current board position as image inputs and outputs a *policy* and a *value*. Based on the evaluated value, MCTS updates the winning probability estimation for each node on the selection path. Meanwhile, the evaluated policy is used in the next selection phase to guide the search process towards more promising moves. By conducting the simulation repeatedly, MCTS progressively refines its estimate of the optimal decision.

### 2.2 Motivation

As introduced earlier, the sequential decision-making process in MCTS leads to high training latencies, and the lack of parallelism bounds the training throughput even when more compute resources are available. In particular, inter-game parallelism, as shown in Figure 1, is mainly employed in scale-out training. On the other hand, intra-decision parallelism – an example of which is parallel MCTS [10] – provides less parallelism opportunities since the parallelism would reduce the strength of MCTS results under the same compute resource bound [11, 14]. These observations motivate us to explore opportunities for exploiting *inter-decision parallelism* to scale the whole training process beyond what can be achieved by state-of-the-art methods.

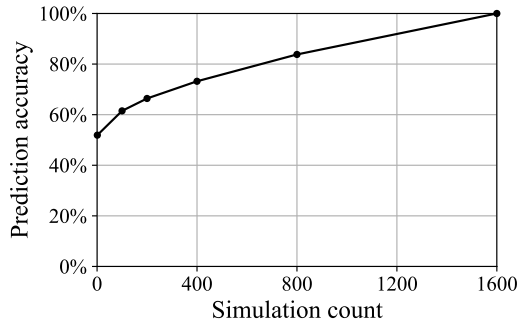

Figure 2: Prediction accuracy when using fewer simulation counts to predict the 1600-simulation's search results.

To show that partial MCTS searches can reasonably *predict* the full MCTS results, we conduct an initial experiment using KataGo [9]. The experiment measures *prediction accuracy* by calculating the fraction of correct predictions made with fewer simulations relative to the complete MCTS with 1600 simulations.[3] As depicted in Figure 2, a partial-search with 800 simulations can achieve an 83.8% prediction accuracy with respect to the full-search, which means that 83.8% of the search results remain unchanged after another 800 simulations. As expected, using fewer simulations leads to lower prediction accuracy. These results provide insights indicating that, more simulations are executed, more confidence we have in the current best candidate. However, several critical board positions could still alter the search outcome in the last few simulations. Consequently, we *cannot* simply speed up MCTS by using fewer simulation counts, as those 16.2% mispredicted search results (from 83.8% to 100%) are vital for improving strength from the 800 to 1600 simulations.

## 3 Related Works

In this section, we discuss the prior works that are most relevant to our paper.

**Parallel MCTS** and other parallel decision tree search algorithms, in a broader sense, belong to the intra-decision parallelism. The parallelization can be implemented in root-level, tree-level, and leaf-level [10]. Several techniques have been proposed to enhance parallel MCTS performance, such as lock-free data structures [15] and virtual loss [10]. Prior study [16] proposed to combine multiple MCTS parallelism, while further study [11, 14] showed that leaf-level parallelism could reduce the performance even only with four workers. Furthermore, prior work [17] proposed to modify the UCT formula to make MCTS more suitable for parallelization. Overall, these previous approaches to "intra-decision" parallelism can potentially be integrated with our "inter-decision" parallelism.

**Speculative parallelism** has also been applied to the $A^\star$ algorithm [18] where nodes are speculatively expanded during the search process. Additionally, prior work [19] speculatively executes MCTS simulation by predicting the next move using a smaller NN. In contrast, our work leverages the anytime algorithm characteristic of MCTS to predict future moves. We focus on addressing the inherent sequential nature of multiple MCTS gameplays that belong to the inter-decision parallelism. On the other hand, speculative parallelism has been explored in non-MCTS contexts and designed at various levels, such as hardware prefetching, thread-level speculation, and compiler loop speculation [20, 21].

## 4 Methodology

In this section, we first present the details of our proposed speculative MCTS approach. Then, we analyze the speculative execution using our analytical models in Section 4.2. Finally, we discuss the synergy effects when combining NN caching with speculative MCTS in Section 4.3.

### 4.1 Speculative MCTS

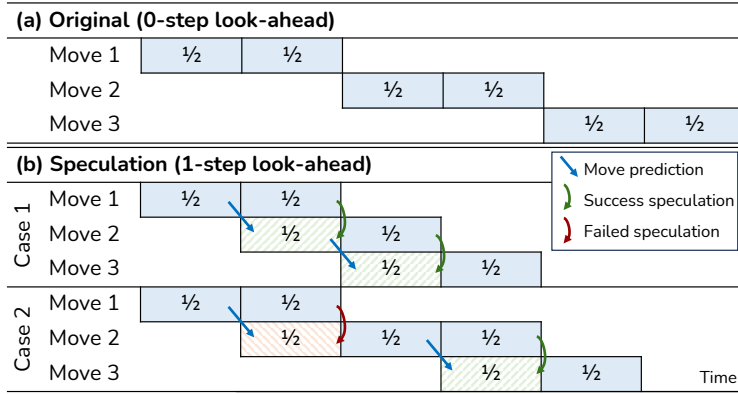

Figure 3: (a) Original execution where each move proceeds sequentially, and (b) Speculation pipeline where the next move is speculatively executed in parallel based on the prediction. Each $^1/_2$ block indicates half of the MCTS computation for a move.

MCTS allows us to obtain partial search results at any point during execution, as it continually updates the search results throughout the process. Hence, we can divide the tree search process into $n$ stages, with each stage handling $^1/_n$ of the total computation. Figure 3 illustrates an example of dividing the tree search process into two stages. In this scenario, when the total number of MCTS simulations for a move is, say, 800, each stage will have 400 simulations. The state-of-the-art MCTS (Figure 3(a)), including parallel MCTS [10], proceeds to the next move *only after* the current tree search is completed.

On the other hand, in our speculative MCTS, we leverage the partial search results from the current pipeline stage to *predict* the next move. In this example, we predict the final outcome of an 800-simulation MCTS using the results from the first 400 simulations. Based on this prediction, we then start the tree search of the next move. Therefore, the next move is *speculatively* executed in parallel with the ongoing tree search of the current move, as depicted in Figure 3(b). Case 1 in Figure 3(b) shows a "successful" speculation where the pipeline execution can continue to proceed, whereas Case 2 illustrates a "failed" speculation, which needs to *flush* the pipeline and *restart* the tree search for the next move (Move 2).

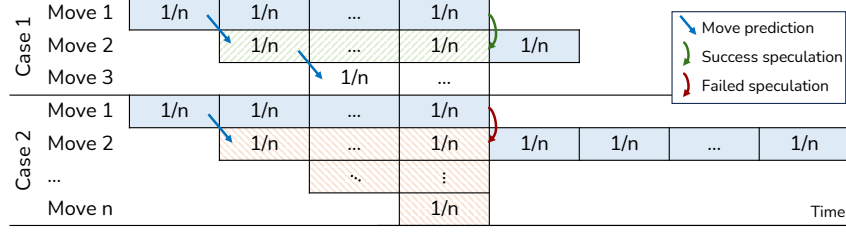

Figure 4: $(n-1)$-step look-ahead speculation contains $n$ pipeline stages. Case 1 shows a successful prediction pipeline, whereas Case 2 illustrates a failed speculation, leading to a pipeline flush.

In general, we can speculate $(n-1)$-step ahead from a move by partitioning the tree search process into $n$ pipeline stages, as shown in Figure 4. We define 0-step look-ahead speculation to be the same as the original execution pipeline, namely, with no speculation occurring when $n = 1$. In Case 1 of Figure 4, successful speculation reduces the latency to $\frac{1}{n}$ of the original move while maintaining the same compute resources. In contrast, in Case 2, a failed speculation results in the same latency as the original move, with increased compute resource demand to $\frac{n+1}{2}$ due to the pipeline flush.[4] Moreover, one could potentially design various speculation strategies to reduce the pipeline flush cost, and we will discuss this further in the following section and in Appendix A.2.

## 4.2 Speculation Analysis

In this section, we present an "analytical approach" to *estimate* the speedup of end-to-end latency and compute resources. As discussed earlier, we have obtained the expected latency required for Case 1 and Case 2 in Figure 4. By knowing the fraction of time the pipeline spends in each case, we can estimate the end-to-end expected speedup as a "weighted average" over the expected latency for each case. Hence, we first model the speculation pipeline (Figure 3(b)) into a finite state machine (Figure 5), and then

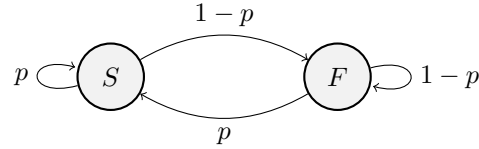

Figure 5: The finite state machine corresponds to the speculation pipeline in Figure 3(b), containing two states: successful ($S$) and failed ($F$) speculation, and the prediction accuracy $p$.

solve for the "steady-state" probability distribution of the pipeline. Since each pipeline case only depends on the previous one, the finite state machine satisfies the *Markov Property*. In general, given a finite state machine with transition probabilities, we solve for the steady-state probability distribution of the finite Markov chain using the following linear equations:

$$\begin{cases} \mathbf{qM} = \mathbf{q} \\ \|\mathbf{q}\| = 1, \end{cases} \tag{1}$$

where $\mathbf{M}$ is the given "transition matrix" and the row vector $\mathbf{q}$ represents "steady-state probability distribution". After solving the equation, we can calculate the expected end-to-end latency by the inner product of $\mathbf{q}$ and a column vector $\mathbf{L}$ that contains the expected latency of each pipeline case. Then, the end-to-end latency is given by $\mathbf{qL}$. Similarly, the compute resource required for each pipeline case is defined as $\mathbf{R}$, and the total amount of required compute resources is $\mathbf{qR}$.

Taking Figure 4 as an example, as previously discussed, we can write down the latency vector as $\mathbf{L} = \left(\frac{1}{n}, 1\right)^T$ and the resource vector as $\mathbf{R} = \left(1, \frac{(n+1)}{2}\right)^T$. Based on the finite state machine in Figure 5, we can determine the expected end-to-end latency and compute resources for the $n$-step

look-ahead speculation and arrive at the following results:

$$p_n := Pr \, (\text{make a success } (n-1)\text{-step look-ahead speculation}) \tag{2}$$

$$\mathbb{E}[latency] = p_n \cdot \frac{1}{n} + (1 - p_n) \cdot 1 \tag{3}$$

$$\mathbb{E}[resource] = p_n \cdot 1 + (1 - p_n) \cdot \frac{n+1}{2}, \tag{4}$$

It is to be emphasized that our analysis is general and applicable to different speculation-based pipelining strategies (see also Appendix A.2), thus enabling us to provision the various pipeline designs beforehand, based on the insights from our analysis.

## 4.3 Synergizing Speculation and Neural Network Caching

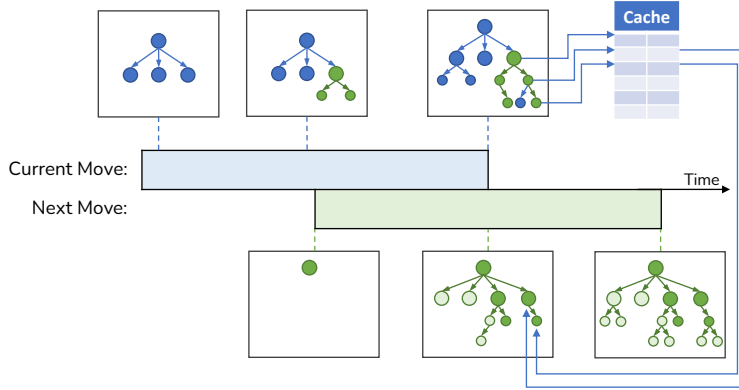

Figure 6: In speculative MCTS, parallel moves *share* cached NN inference results, which accelerate the execution mutually.

The NN inference results at each tree node can typically be *cached* in a hash table, with the NN input being the hash key.[5] The costly NN inference can thus be alleviated during tree node evaluation upon a cache hit. This caching technique also applies to our speculative MCTS. In the original MCTS, different moves access the NN cache in sequence according to the gameplay order. As a result, the current move can only benefit from the inference results cached by previous moves and the current move itself. Unlike the original MCTS, speculative MCTS allows both the current and the next moves to *simultaneously* access the NN cache. Thus, regardless of which move performs the node inference first, the subsequent tree search can benefit from the cached NN results. For example, the dark green nodes in Figure 6 represent the *shared* positions between the current and next moves, which can potentially accelerate the execution mutually via the shared NN cache. In essence, *both* the current and speculated moves can take advantage of the NN inference results, and thus, the cache hit rate is expected to increase as the speculative look-ahead step increases. Furthermore, even if the speculated moves are mispredicted and subsequently flushed, their NN inference results can still serve future moves and are likely to contribute to the hits in the main sequence of MCTS. Therefore, we expect the synergy between "speculation" and "caching" to provide more speedups than employing each technique individually.

Moreover, based on the introduced analysis, we can estimate the latency as follows:

$$\mathbb{E}[latency \ w/\$] = \mathbb{E}[latency \ w/o \ \$] \times (1 - hr) + t_0, \tag{5}$$

where $hr$ denotes the NN cache hit rate, $t_0$ is the time to access the hash table, which is significantly shorter than performing an NN inference, and $\mathbb{E}[latency \ w/o \ \$]$ can be obtained from Equation 3.

## 5 Evaluations

In this section, we first provide our experimental setting in Section 5.1. Then, we evaluate the training latency in Section 5.2, and study the NN cache hit rate in Section 5.3, to provide insights into our

observed speedups. Finally, we evaluate the performance of end-to-end training and compare our proposed method against state-of-the-art approaches in Section 5.4.

## 5.1 Experimental Setting

We evaluate our speculative MCTS approach on 9x9 NoGo [24], 9x9 Go, and 19x19 Go games using different NN sizes. The NN models consist of ResNet blocks [25], denoted as bXcY, where X represents the number of blocks and Y is the number of channels per block. The input features for NoGo and Go games follow the configurations established in prior works [26, 27]. Additionally, all models are trained from randomly initialized weights, and the self-play simulation is 800.

Our experiments are conducted on a cluster that consists of 8 NVIDIA V100 GPUs per node. We train four NN models, (9x9 go, b5c64), (9x9 go, b10c128), (9x9 nogo, b5c64) and (19x19 go, b6c96), and each model is trained for 1400 V100-hours. For the latency evaluation in Sections 5.2 and 5.3, we run the experiments on one GPU and measure the latency by averaging the results over 100,000 game moves. For end-to-end evaluation in Section 5.4, we use 5 GPU nodes for both two runs. Moreover, we perform a strength evaluation against the state-of-the-art method, using an extra 280 V100-hours. In all our evaluations, the batch sizes for training and inference are set to 1024 and 24, respectively.

## 5.2 Evaluation of Training Latency

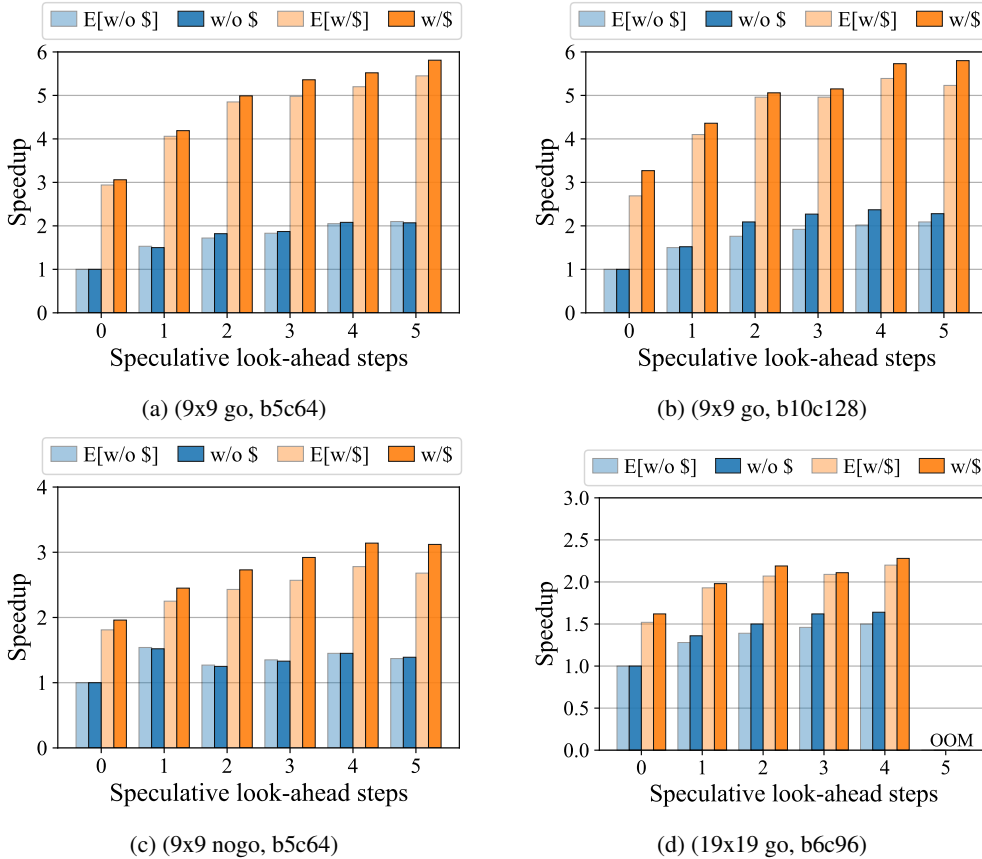

(a) (9x9 go, b5c64)

(b) (9x9 go, b10c128)

(c) (9x9 nogo, b5c64)

(d) (19x19 go, b6c96)

Figure 7: Speedup for speculative MCTS across various speculative look-ahead steps, with each subfigure representing a different (game, NN size) setting. OOM denotes "out-of-memory" evaluation.

This section evaluates the per-iteration self-play latency with different configurations of speculation look-ahead steps and NN caching. Figure 7 shows the speedup of our speculative MCTS compared to the baseline (original) MCTS, which is executed *without* speculation (i.e., 0 look-ahead step). Based on our analysis described in Section 4.2, the light blue and orange bars indicate the expected speedup

without NN cache (denoted as E[w/o $]) and with NN cache (denoted as E[w/$]), respectively, whereas the dark blue and orange bars show evaluated speedups without and with NN cache, respectively.

For the performance without the NN cache (blue bars), the results indicate that all speculation cases are faster than the baseline. Among the results, the speculation with four look-ahead steps achieves a speedup of 2.37 over the baseline in (9x9 go, b10c128). We also observe that, in general, performance improves when the speculative look-ahead step is increased, but the speedup does not improve significantly. The reason is that, as the speculative look-ahead step increases, we predict the next move with MCTS partial results of fewer simulations. As a result, the accuracy of the speculative prediction might be reduced, leading to less speedup improvement. For the comparison of the estimated speedup and the evaluated speedup, Equation 3 gives a close estimation of 22.2 RMSE (root mean square error) across all four settings.

For the performance with NN cache (orange bars), the speedup increases as the speculative look-ahead step increases, and we achieve a speedup of up to 5.81 with five speculative look-ahead steps over the baseline MCTS in (9x9 go, b5c64). Furthermore, we observe that the values of evaluated speedups are greater than those of the estimated speedups among all four settings. This is because our analytical modeling (Equation 5) does *not* take the synergistic effects into account. Therefore, *combining* speculation and NN cache would, in general, lead to a better speedup. Overall, the speculation with the NN cache offers at least 1.25 speedup over the baseline (0 look-ahead step) with the NN cache, and achieves 2.1 times improvement, on average, compared to the speculation without the cache.

## 5.3  Evaluation of Cache Hit Rate

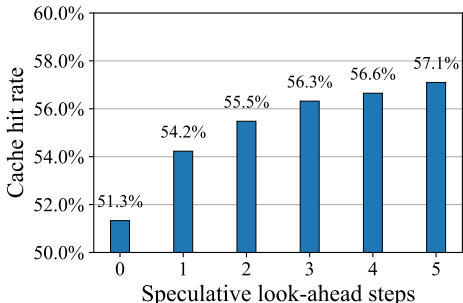

Figure 8: NN cache hit rate across different speculative look-ahead steps.

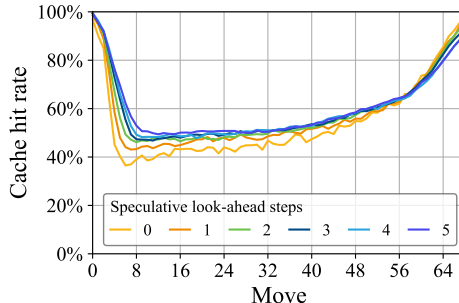

Figure 9: NN cache hit rate for each move across various speculative look-ahead steps.

In this section, we dissect the speculative MCTS speedup in (nogo, b5c64) by examining the contribution of the NN cache hit rate to the overall performance. Figure 8 indicates that, as the speculative look-ahead step is increased, the NN cache hit rate also increases, aligning with our discussion in Section 4.3. To further investigate the source of this improvement, we measure the cache hit rates of each move during self-play. The result in Figure 9 reveals that the cache hit rates are high at the beginning and the end of the game, but drop in the middle of the game. In general, the NN cache hits come from two sources: (1) intra-game hits, which leverage temporal locality as moves within a game often overlap in multiple MCTS's tree nodes, and (2) inter-game hits, which are akin to caching an opening book for the game and the concept is similar to the "joseki" in Go. Hence, the result matches our intuition that common opening board positions are shared across different games while many endgame moves recur in tree searches within the same game, thus leading to high cache hit rates. During the middle of the game, MCTS explores more board positions compared to the opening and endgame, and thus, move decisions are more divergent, leading to lower cache hit rates. In this case, our speculative MCTS can provide more intra-game hit opportunities, which improve the cache hit rates. This can be observed at move 7 in Figure 9, where the cache hit rate is increased from 36% with no speculation to 62% with 5-step ahead speculation—an improvement of 26%.

## 5.4  Evaluation of End-to-End Training

In this section, we evaluate the performance of our speculative MCTS against the state-of-the-art method. The KataGo [9] model and its training are based on settings in the public repository [28].

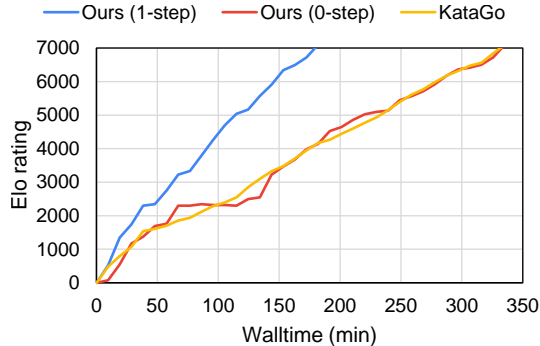

Figure 10: End-to-end training evaluation against the state-of-the-art method, KataGo.

Since our training settings described in Section 5.1 are different from those of prior works, we extrapolated their training time. Furthermore, for each model iteration, we play 400 games against KataGo's model to evaluate the Elo rating [29], establishing a 95% confidence interval for the win rate within $\pm 5\%$.

Figure 10 shows the comparison result. The results show that the training performance of MCTS without speculation (0-step) aligns with the state-of-the-art Go program, KataGo. Moreover, our speculative MCTS with 1-step look-ahead speculation achieves $1.91\times$ speedup over KataGo when reaching 7,000 Elo rating. This performance also aligns with the per-iteration evaluation result in Figure 7d. Though we only evaluated our method with only 1-step look-ahead speculation since the training is resource-consuming, we expect the end-to-end training will follow the same trend in Figure 7 and continue to scale up further when more compute resources are available.

# 6 Discussion

## 6.1 Design Alternatives for Inter-Decision Parallelism

To achieve inter-decision parallelism in MCTS, we need to break the "sequential" nature of the gameplay, and this typically comes with *predicting* the next move. Naively, we could store all transitions of the game states so that when a new move is started, we can immediately execute the next move or even several future moves in parallel. However, in practice, this is not feasible due to the complexity of the game. On the other hand, in contrast to the prior work [19], which needs an extra NN for speculation prediction and incurs additional cost, our method leverages the anytime algorithm characteristics of MCTS, and this naturally leads us to our proposed approach.

## 6.2 Design Choices in Speculative MCTS

The design of speculative MCTS involves (1) improving prediction accuracy, and (2) reducing the speculation cost. For the first design dimension, we investigated the number of steps to *look ahead* in the speculation. Our evaluation shows that, even with only one look-ahead step, we can speed up the training up to $3\times$. Furthermore, the training can benefit from more speculative look-ahead steps when adding more computing resources. For the second design dimension, we integrate NN caching, and more importantly, our analytical method supports general design spaces, which can analyze potential speculation strategies beforehand.

## 6.3 Limitation of Speculative MCTS

Speculative MCTS introduces a new dimension of parallelism. This approach is more beneficial when computing resources are abundant. When the game duration or the total MCTS simulation is short, the performance gain would be limited since the performance improvement comes from looking ahead into future moves. Fortunately, game applications typically utilize high simulation counts or require a lengthy sequential decision process and, thus, can benefit from our approach.

# 7   Concluding Remarks

In this paper, we introduce and experimentally evaluate **speculative MCTS** as a novel "inter-decision" parallelism maximization strategy, to reduce the time-to-solution in AlphaZero training. We provide insights into the anytime characteristic of MCTS, which allows us to *predict* future moves with partial search results and *speculatively execute* the future moves with the current move *in parallel*. We further introduce the NN cache and analyze the synergy between the caching mechanism and speculation. Additionally, we propose an analytical model that can be used to analyze and evaluate different speculation strategies/implementations. Our empirical results indicate that the proposed speculative MCTS can accelerate training by up to $5.81\times$ in 9x9 Go. The end-to-end training results also show a $1.91\times$ speedup on 19x19 Go game training, compared to a state-of-the-art Go program, KataGo. Finally, our speculative MCTS can combine with other intra-decision parallelism approaches, such as parallel MCTS [10], to enable further performance improvement.

## Acknowledgments and Disclosure of Funding

This research was funded in part by National Science and Technology Council of Taiwan project number NSTC113-2221-E-001-019. We thank the Taiwan National Center for High-performance Computing (NCHC) for providing computational and storage resources. Lastly, we would like to thank Meng-Yu Tsai for the intellectual discussions.

## Footnotes

[1]AlphaZero's training data per parameter can be estimated by (21M games $\times$ 150 positions/game) / (24M parameters) = 131 positions/parameter. In comparison, LLM's training sample per parameter is about 20 according to the Chinchilla scaling law [8].

[2]An algorithm capable of delivering a "valid" result, even if it is halted before completion.

[3]Similar evaluations and observations can be done with different MCTS simulations (see also Appendix A.1).

[4] In this context, compute resource is the sum of pipeline flush cost (orange stages in Case 2) and recomputation cost $= \frac{1}{n} + \frac{2}{n} + \ldots + \frac{(n-1)}{n} + 1 = \frac{(n+1)}{2}$.

[5]To avoid the GHI problem, we only allow two identical board positions to share the same NN cache when they have the same history, particularly for the Go game [22, 23].

# References

[1] R. Coulom, "Efficient selectivity and backup operators in monte-carlo tree search," in *Computers and Games: 5th International Conference, CG 2006, Turin, Italy, May 29-31, 2006. Revised Papers 5*, pp. 72–83, Springer, 2007.

[2] L. Kocsis and C. Szepesvári, "Bandit based monte-carlo planning," in *Machine Learning: ECML 2006: 17th European Conference on Machine Learning Berlin, Germany, September 18-22, 2006 Proceedings 17*, pp. 282–293, Springer, 2006.

[3] M. H. Segler, M. Preuss, and M. P. Waller, "Planning chemical syntheses with deep neural networks and symbolic ai," *Nature*, vol. 555, no. 7698, pp. 604–610, 2018.

[4] S. S. SV, J. N. Law, C. E. Tripp, D. Duplyakin, E. Skordilis, D. Biagioni, R. S. Paton, and P. C. St. John, "Multi-objective goal-directed optimization of de novo stable organic radicals for aqueous redox flow batteries," *Nature Machine Intelligence*, vol. 4, no. 8, pp. 720–730, 2022.

[5] H. Qian, C. Lin, D. Zhao, S. Tu, and L. Xu, "Alphadrug: protein target specific de novo molecular generation," *PNAS Nexus*, vol. 1, no. 4, p. pgac227, 2022.

[6] D. Erikawa, N. Yasuo, and M. Sekijima, "Mermaid: an open source automated hit-to-lead method based on deep reinforcement learning," *Journal of Cheminformatics*, vol. 13, pp. 1–10, 2021.

[7] D. Silver, T. Hubert, J. Schrittwieser, I. Antonoglou, M. Lai, A. Guez, M. Lanctot, L. Sifre, D. Kumaran, T. Graepel, *et al.*, "A general reinforcement learning algorithm that masters chess, shogi, and go through self-play," *Science*, vol. 362, no. 6419, pp. 1140–1144, 2018.

[8] J. Hoffmann, S. Borgeaud, A. Mensch, E. Buchatskaya, T. Cai, E. Rutherford, D. d. L. Casas, L. A. Hendricks, J. Welbl, A. Clark, *et al.*, "Training compute-optimal large language models," *arXiv preprint arXiv:2203.15556*, 2022.

[9] D. J. Wu, "Accelerating self-play learning in go," *arXiv preprint arXiv:1902.10565*, 2019.

[10] G. M. B. Chaslot, M. H. Winands, and H. J. van Den Herik, "Parallel monte-carlo tree search," in *Computers and Games: 6th International Conference, CG 2008, Beijing, China, September 29-October 1, 2008. Proceedings 6*, p. 60–71, Springer, 2008.

[11] Y. Soejima, A. Kishimoto, and O. Watanabe, "Evaluating root parallelization in go," *IEEE Transactions on Computational Intelligence and AI in Games*, vol. 2, no. 4, pp. 278–287, 2010.

[12] M. Boddy and T. L. Dean, *Solving time-dependent planning problems*. Brown University, Department of Computer Science, 1989.

[13] A. Mandhane, A. Zhernov, M. Rauh, C. Gu, M. Wang, F. Xue, W. Shang, D. Pang, R. Claus, C.-H. Chiang, *et al.*, "Muzero with self-competition for rate control in vp9 video compression," *arXiv preprint arXiv:2202.06626*, 2022.

[14] A. Bourki, G. Chaslot, M. Coulm, V. Danjean, H. Doghmen, J.-B. Hoock, T. Hérault, A. Rimmel, F. Teytaud, O. Teytaud, *et al.*, "Scalability and parallelization of monte-carlo tree search," in *Computers and Games: 7th International Conference, CG 2010, Kanazawa, Japan, September 24-26, 2010, Revised Selected Papers 7*, pp. 48–58, Springer, 2011.

[15] S. A. Mirsoleimani, H. J. van den Herik, A. Plaat, and J. Vermaseren, "A lock-free algorithm for parallel mcts.," in *ICAART (2)*, pp. 589–598, 2018.

[16] Y. Meng, Q. Wang, T. Zu, and V. Prasanna, "Accelerating deep neural network guided mcts using adaptive parallelism," in *Proceedings of the SC'23 Workshops of The International Conference on High Performance Computing, Network, Storage, and Analysis*, p. 766–769, 2023.

[17] A. Liu, J. Chen, M. Yu, Y. Zhai, X. Zhou, and J. Liu, "Watch the unobserved: A simple approach to parallelizing monte carlo tree search," *arXiv preprint arXiv:1810.11755*, 2018.

[18] M. Bakhshalipour, M. Qadri, D. Guri, S. B. Ehsani, M. Likhachev, and P. B. Gibbons, "Runahead a*: Speculative parallelism for a* with slow expansions," in *Proceedings of the International Conference on Automated Planning and Scheduling*, vol. 33, pp. 31–41, 2023.

[19] J. Kim, B. Kang, and H. Cho, "Specmcts: Accelerating monte carlo tree search using speculative tree traversal," *IEEE Access*, vol. 9, p. 142195–142205, 2021.

[20] J. F. Martinez and J. Torrellas, "Speculative synchronization: Applying thread-level speculation to explicitly parallel applications," *ACM SIGOPS Operating Systems Review*, vol. 36, no. 5, pp. 18–29, 2002.

[21] A. Estebanez, D. R. Llanos, and A. Gonzalez-Escribano, "A survey on thread-level speculation techniques," *ACM Computing Surveys (CSUR)*, vol. 49, no. 2, pp. 1–39, 2016.

[22] A. Kishimoto and M. Müller, "A general solution to the graph history interaction problem," in *Proceedings of the 19th national conference on Artifical intelligence*, pp. 644–649, 2004.

[23] A. J. Palay, *Searching with probabilities*. Pitman London, 1985.

[24] M. Müller, "Nogo history and competitions." "`https://webdocs.cs.ualberta.ca/~mmueller/nogo/history.html`", 2015.

[25] K. He, X. Zhang, S. Ren, and J. Sun, "Deep residual learning for image recognition," in *Proceedings of the IEEE conference on computer vision and pattern recognition*, pp. 770–778, 2016.

[26] T. Cazenave, Y.-C. Chen, G.-W. Chen, S.-Y. Chen, X.-D. Chiu, J. Dehos, M. Elsa, Q. Gong, H. Hu, V. Khalidov, *et al.*, "Polygames: Improved zero learning," *ICGA Journal*, vol. 42, no. 4, pp. 244–256, 2020.

[27] D. Silver, A. Huang, C. J. Maddison, A. Guez, L. Sifre, G. Van Den Driessche, J. Schrittwieser, I. Antonoglou, V. Panneershelvam, M. Lanctot, *et al.*, "Mastering the game of go with deep neural networks and tree search," *nature*, vol. 529, no. 7587, pp. 484–489, 2016.

[28] D. J. Wu, "Networks for kata1." "`https://katagotraining.org/networks/`", 2020.

[29] R. Coulom, "Computing "elo ratings" of move patterns in the game of go," *ICGA journal*, vol. 30, no. 4, pp. 198–208, 2007.

[30] A. Meurer, C. P. Smith, M. Paprocki, O. Čertík, S. B. Kirpichev, M. Rocklin, A. Kumar, S. Ivanov, J. K. Moore, S. Singh, *et al.*, "Sympy: symbolic computing in python," *PeerJ Computer Science*, vol. 3, p. e103, 2017.

# A    Appendix / Supplemental Material

## A.1    Scaling of the Prediction Accuracy

In addition to Figure 2, Figure 11 further evaluates the prediction accuracy for simulations ranging from 50 to 3200. In general, the prediction accuracy follows a consistent trend across all evaluations. These results highlight that prediction accuracy scales effectively among different simulations and further suggest that our speculative MCTS approach is scalable across various numbers of simulations.

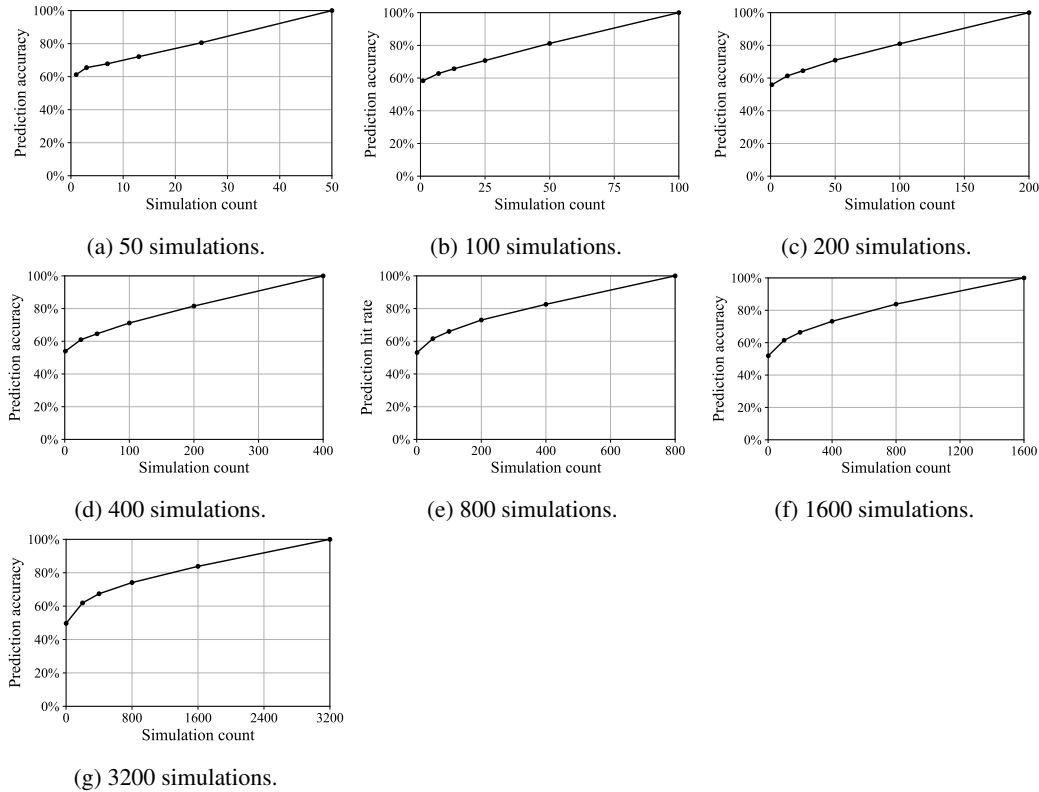

(a) 50 simulations.      (b) 100 simulations.      (c) 200 simulations.

(d) 400 simulations.      (e) 800 simulations.      (f) 1600 simulations.

(g) 3200 simulations.

Figure 11: Prediction accuracy using 50 to 3200 simulations as the full MCTS simulations.

## A.2    Alternative Speculation Strategy

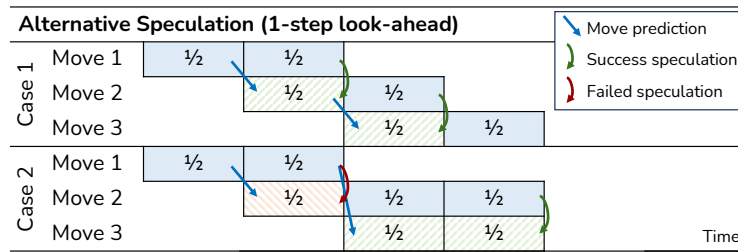

Figure 12: Alternative speculation pipeline with 1 look-ahead step. Specifically, in Case 2, the speculation starts immediately after a pipeline flush.

As discussed earlier in Section 4.1, when the speculation fails, the pipeline is flushed and pipeline "bubbles" are introduced. This section explores an alternative design: instead of inserting pipeline bubbles, the speculation is immediately started based on the prediction from the prior tree search result. For instance, in Figure 12, Case 2 shows that, when the speculation for Move 2 fails, the pipeline can still speculate Move 3 using the prediction from the previous move (Move 1).

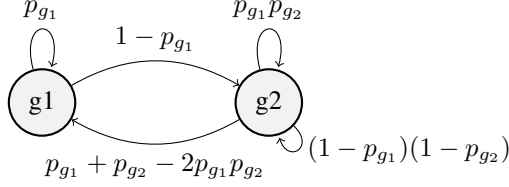 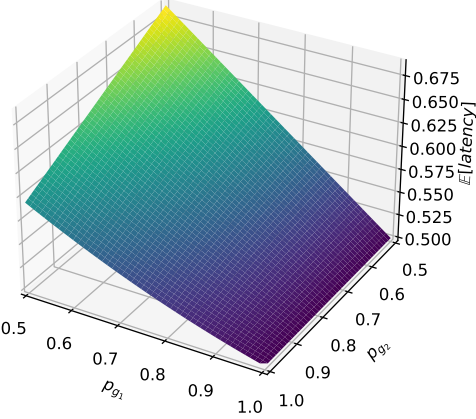

Figure 13: The finite state machine corresponds to Figure 12, containing two states: g1 (guess 1-step ahead) and g2 (guess 2-step ahead), and the prediction accuracy $p_{g_1}, p_{g_2}$.

Figure 14: Visualization for the expected latency of the speculation pipeline in Figure 12, which is a function of prediction accuracy $p_{g_1}$ and $p_{g_2}$.

Similar to Figure 5, we first formulate the finite state machine, shown in Figure 13, based on the speculation pipeline cases in Figure 12, where $p_{g_1} := p_1$ as defined in Equation 2 and $p_{g_2}$ is the probability to make a successful prediction using the prior tree search result. Specifically, g1 has $p_{g_1}$ probability to remain successful prediction, thus having $(1 - p_{g_1})$ probability to transit to the g2 state. On the other hand, g2 remains a successful prediction when both pipeline stages make successful or failed prediction, corresponding to the probabilities of $p_{g_1} p_{g_2}$ and $(1 - p_{g_1})(1 - p_{g_2})$, respectively. As a result, we can obtain the transition matrix:

$$\mathbf{M} = \begin{pmatrix} p_{g_1} & 1 - p_{g_1} \\ p_{g_1} + p_{g_2} - 2p_{g_1}p_{g_2} & 1 - p_{g_1} - p_{g_2} + 2p_{g_1}p_{g_2} \end{pmatrix}. \tag{6}$$

Following Equation 1, we can solve for the steady-state probability distribution $\mathbf{q}$ via a symbolic solver like SymPy [30], which gives us:

$$\mathbf{q} = \begin{pmatrix} \frac{2p_{g_1}p_{g_2} - p_{g_1} - p_{g_2}}{2p_{g_1}p_{g_2} - p_{g_2} - 1} & \frac{p_{g_1} - 1}{2p_{g_1}p_{g_2} - p_{g_2} - 1} \end{pmatrix}. \tag{7}$$

Further, we can obtain the expected latency for each state based on Figure 12 and Figure 13:

$$\mathbb{E}[\text{latency of g1}] = p_{g_1} \cdot \frac{1}{2} + (1 - p_{g_1}) \cdot 0 = \frac{p_{g_1}}{2} \tag{8}$$

$$\mathbb{E}[\text{latency of g2}] = p_{g_1}p_{g_2} \cdot \frac{1}{2} + (1 - p_{g_1})(1 - p_{g_2}) \cdot 1 + (p_{g_1} + p_{g_2} - 2p_{g_1}p_{g_2}) \cdot \frac{3}{4}, \tag{9}$$

and thus, $\mathbf{L} = (\mathbb{E}[\text{latency of g1}], \mathbb{E}[\text{latency of g2}])^T$. Finally, based on the inner product of $\mathbf{q}$ and $\mathbf{L}$, we arrive at the expected latency for the speculation pipeline:

$$\frac{2p_{g_1}^3 p_{g_2} - p_{g_1}^3 + 2p_{g_1}^2 p_{g_2}^2 - 8p_{g_1}^2 p_{g_2} + 2p_{g_1}^2 - 3p_{g_1}p_{g_2}^2 + 11p_{g_1}p_{g_2} + p_{g_1} + p_{g_2}^2 - 3p_{g_2} - 4}{4 \cdot (2p_{g_1}p_{g_2} - p_{g_2} - 1)}. \tag{10}$$

Figure 14 shows the visualization for $\mathbb{E}[latency]$ in Equation 10, which is a function of prediction accuracies $p_{g_2}$ and $p_{g_2}$. In the visualization, we can clearly see that, when all predictions are successful, i.e., $p_{g_1} = 1$ and $\mathbb{E}[latency] = 0.5$, we reach the ideal $2\times$ speedup, and this matches our intuition. When we fix $p_{g_1}$, the higher the prediction accuracy $p_{g_2}$, the lower the overall latency. Similarly, when we fix $p_{g_2}$, the higher the prediction accuracy $p_{g_1}$, the lower the expected latency. Moreover, suppose we obtain $(p_{g_1}, p_{g_2}) = (83.8\%, 79.8\%)$ from a profiling similar to Figure 2 in the Go game. In this case, we see the expected latency speedup to be 1.84, while the method in Section 4.1 has a lower speedup of 1.72. However, the speedup is improved at the cost of complexity in the implementation. Nevertheless, our analytical model in Section 4.2 can be successfully adapted/applied to various speculative MCTS designs.

